# Application of Variational Bayesian Approach to Speech Recognition

**Shinji Watanabe, Yasuhiro Minami, Atsushi Nakamura and Naonori Ueda**
NTT Communication Science Laboratories, NTT Corporation
2-4, Hikaridai, Seika-cho, Soraku-gun, Kyoto, Japan
{watanabe,minami,ats,ueda}@cslab.kecl.ntt.co.jp

## Abstract

In this paper, we propose a Bayesian framework, which constructs shared-state triphone HMMs based on a variational Bayesian approach, and recognizes speech based on the Bayesian prediction classification; variational Bayesian estimation and clustering for speech recognition (VBEC). An appropriate model structure with high recognition performance can be found within a VBEC framework. Unlike conventional methods, including BIC or MDL criterion based on the maximum likelihood approach, the proposed model selection is valid in principle, even when there are insufficient amounts of data, because it does not use an asymptotic assumption. In isolated word recognition experiments, we show the advantage of VBEC over conventional methods, especially when dealing with small amounts of data.

## 1 Introduction

A statistical modeling of spectral features of speech (acoustic modeling) is one of the most crucial parts in the speech recognition. In acoustic modeling, a triphone-based hidden Markov model (triphone HMM) has been widely employed. The triphone is a context dependent phoneme unit that considers both the preceding and following phonemes. Although the triphone enables the precise modeling of spectral features, the total number of triphones is too large to prepare sufficient amounts of training data for each triphone. In order to deal with the problem of data insufficiency, an HMM state is usually shared among multiple triphone HMMs, which means the amount of training data per state inflates. Such shared-state triphone HMMs (SST-HMMs) can be constructed by successively clustering states based on the phonetic decision tree method [4] [7]. The important practical problem that must be solved when constructing SST-HMMs is how to optimize the total number of shared states adaptively to the amounts of available training data. Namely, maintaining the balance between model complexity and training data size is quite important for high generalization performance.

The maximum likelihood (ML) is inappropriate as a model selection criterion since ML increases monotonically as the number of states increases. Some heuristic thresholding is therefore necessary to terminate the partitioning. To solve this problem, the Bayesian information criterion (BIC) and minimum description length (MDL) criterion have been

employed to determine the tree structure of SST-HMMs [2] [5] [1]. However, since the BIC/MDL is based on an asymptotic assumption, it is invalid in principle when the number of training data is small because of the failure of the assumption.

In this paper, we present a practical method within the Bayesian framework for estimating posterior distributions over parameters and selecting an appropriate model structure of SST-HMMs (clustering triphone HMM states) based on a variational Bayesian (VB) approach, and recognizing speech based on the Bayesian prediction classification: variational Bayesian estimation and clustering for speech recognition (VBEC). Unlike the BIC/MDL, VB does not assume asymptotic normality, and it is therefore applicable in principle, even when there are insufficient data. The VB approach has been successfully applied to model selection problems, but mainly for relatively simple mixture models [1] [3] [6] [8]. Here, we try to apply VB to SST-HMMs with more a complex model structure than the mixture model and evaluate the effectiveness through a large-scale real speech recognition experiment.

## 2   Variational Bayesian framework

First, we briefly review the VB framework. Let $\boldsymbol{O}$ be a given data set. In the Bayesian approach we are interested in posterior distributions over model parameters, $p(\Theta|\boldsymbol{O}, m)$, and the model structure, $p(m|\boldsymbol{O})$. Here, $\Theta$ is a set of model parameters and $m$ is an index of the model structure. Let us consider a general probabilistic model with latent variables. Let $Z$ be a set of latent variables. Then the model with a fixed model structure $m$ can be defined by the joint distribution $p(\boldsymbol{O}, Z|\Theta, m)$.

In VB, variational posteriors $q(\Theta|\boldsymbol{O}, m)$, $q(Z|\boldsymbol{O}, m)$, and $q(m|\boldsymbol{O})$ are introduced to approximate the true corresponding posteriors. The optimal variational posteriors over $\Theta$ and $Z$, and the appropriate model structure that maximizes the optimal $q(m|\boldsymbol{O})$ can be obtained by maximizing the following objective function:

$$\mathcal{F}_m[q] \;=\; \left\langle \log \frac{p(\boldsymbol{O}, Z|\Theta, m)p(\Theta|m)}{q(Z|\boldsymbol{O}, m)q(\Theta|\boldsymbol{O}, m)} \right\rangle_{q(Z|\boldsymbol{O}, m), q(\Theta|\boldsymbol{O}, m)}, \tag{1}$$

w.r.t. $q(\Theta|\boldsymbol{O}, m), q(Z|\boldsymbol{O}, m)$, and $m$. Here $\langle f(x)\rangle_{p(x)}$ denotes the expectation of $f(x)$ w.r.t. $p(x)$. $p(\Theta|m)$ is a prior distribution. This optimization can be effectively performed by an EM-like iterative algorithm (see [1] for the details).

## 3   Applying a VB approach to acoustic models

### 3.1   Output distributions and prior distributions

We attempt to apply a VB approach to a left-to-right HMM, which has been widely used to represent a phoneme unit in acoustic models for speech recognition, as shown in Figure 1. Let $\boldsymbol{O} = \{\boldsymbol{O}^t \in \mathcal{R}^D : t = 1, ..., T\}$ be a sequential data set for a phoneme unit. The output distribution in an HMM is given by

$$p(\boldsymbol{O}, S, V|\Theta, m) = \prod_{t=1}^{T} a_{s^{t-1}s^t} c_{s^t v^t} b_{s^t v^t}(\boldsymbol{O}^t), \tag{2}$$

where $S$ is a set of sequences of hidden states, $V$ is a set of sequences of Gaussian mixture components, and $s^t$ and $v^t$ denote the state and mixture components at time $t$. $S$ and $V$ are sets of discrete latent variables that correspond to $Z$ mentioned above. $a_{ij}$ denotes the state

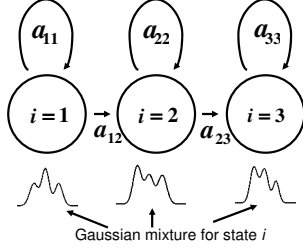

Figure 1: Hidden Markov model for each phoneme unit. A state is represented by the Gaussian mixture distribution below the state. There are three states and three Gaussian components in this figure.

Gaussian mixture for state $i$

transition probability from state $i$ to state $j$, and $c_{jk}$ is the $k$-th weight factor of the Gaussian mixture for state $j$. $b_{jk}(= \mathcal{N}(\boldsymbol{O}^t|\boldsymbol{\mu}_{jk}, \Sigma_{jk}))$ denotes the Gaussian distribution with mean vector $\boldsymbol{\mu}_{jk}$ and covariance $\Sigma_{jk}$. $\Theta = \{a_{ij}, c_{jk}, \boldsymbol{\mu}_{jk}, \Sigma_{jk}^{-1}|i, j = 1, ..., J, k = 1, ..., L\}$ is a set of model parameters. $J$ denotes the number of states in an HMM and $L$ denotes the number of Gaussian components in a state. In this paper, we restrict covariance matrices in the Gaussian distribution to diagonal ones. The conjugate prior distributions are assumed to be as follows:

$$
\begin{aligned}
p(\Theta|m) &= \prod_{i,j,k} \mathcal{D}(\{a_{ij'}\}_{j'=1}^J|\phi^0)\mathcal{D}(\{c_{jk'}\}_{k'=1}^L|\varphi^0) \\
&\times \mathcal{N}(\boldsymbol{\mu}_{jk}|\boldsymbol{\nu}_{jk}^0, (\xi^0)^{-1}\Sigma_{jk})\prod_{d=1}^D \mathcal{G}(\Sigma_{jk,d}^{-1}|\eta^0, R_{jk,d}^0).
\end{aligned} \tag{3}
$$

$\Phi^0 = \{\phi^0, \varphi^0, \boldsymbol{\nu}_{jk}^0, \xi^0, \eta^0, R_{jk}^0\}$ is a set of hyperparameters. We assume the hyperparameters are constants. In Eq.(3), $\mathcal{D}$ denotes a Dirichlet distribution and $\mathcal{G}$ denotes a gamma distribution.

### 3.2 Optimal variational posterior distribution $\tilde{q}(\Theta|\boldsymbol{O}, m)$

From the output distributions and prior distributions in section 3.1, the optimal variational posterior distribution $\tilde{q}(\Theta|\boldsymbol{O}, m)$ can be obtained as:

$$
\begin{aligned}
\tilde{q}(\{a_{ij}\}_{j=1}^J|\boldsymbol{O}, m) &= \mathcal{D}(\{a_{ij}\}_{j=1}^J|\{\tilde{\phi}_{ij}\}_{j=1}^J) \\
\tilde{q}(\{c_{jk}\}_{k=1}^L|\boldsymbol{O}, m) &= \mathcal{D}(\{c_{jk}\}_{k=1}^L|\{\tilde{\varphi}_{jk}\}_{k=1}^L) \\
\tilde{q}(b_{jk}|\boldsymbol{O}, m) &= \mathcal{N}(\boldsymbol{\mu}_{jk}|\tilde{\boldsymbol{\nu}}_{jk}, \tilde{\xi}_{jk}^{-1}\Sigma_{jk})\prod_{d=1}^D \mathcal{G}(\Sigma_{jk,d}^{-1}|\tilde{\eta}_{jk}, \tilde{R}_{jk,d}),
\end{aligned} \tag{4}
$$

$\tilde{\Phi} \equiv \{\tilde{\phi}, \tilde{\varphi}, \tilde{\boldsymbol{\nu}}_{jk}, \tilde{\xi}, \tilde{\eta}, \tilde{R}_{jk}\}$ is a set of posterior distribution parameters defined as:

$$
\tilde{\phi}_{ij} = \phi^0 + \tilde{\gamma}_{ij}, \ \tilde{\varphi}_{jk} = \varphi^0 + \tilde{\zeta}_{jk}, \ \tilde{\xi}_{jk} = \xi^0 + \tilde{\zeta}_{jk}, \ \tilde{\boldsymbol{\nu}}_{jk} = \left(\xi^0\boldsymbol{\nu}_{jk}^0 + \sum_{t=1}^T \tilde{\zeta}_{jk}^t\boldsymbol{O}^t\right)/\tilde{\xi}_{jk},
$$

$$
\tilde{\eta}_{jk} = \eta^0 + \tilde{\zeta}_{jk}, \ \tilde{R}_{jk,d} = R_{jk,d}^0 + \xi^0(\nu_{jk,d}^0 - \tilde{\nu}_{jk,d})^2 + \sum_{t=1}^T \tilde{\zeta}_{jk}^t(O_d^t - \tilde{\nu}_{jk,d})^2. \tag{5}
$$

$\tilde{\Phi}$ is composed of $\tilde{\gamma}_{ij}^t \equiv \tilde{q}(s^t = i, s^{t+1} = j|\boldsymbol{O}, m)$, $\tilde{\gamma}_{ij} \equiv \Sigma_{t=1}^T\tilde{\gamma}_{ij}^t$, $\tilde{\zeta}_{jk}^t \equiv \tilde{q}(s^t = j, v^t = k|\boldsymbol{O}, m)$ and $\tilde{\zeta}_{jk} \equiv \Sigma_{t=1}^T\tilde{\zeta}_{jk}^t$. $\tilde{\gamma}_{ij}^t$ denotes the transition probability from state $i$ to state $j$ at time $t$. $\tilde{\zeta}_{jk}^t$ denotes the occupation probability on mixture component $k$ in state $j$ at time $t$.

### 3.3 Optimal variational posterior distribution $\tilde{q}(S, V|\boldsymbol{O}, m)$

From the output distributions and prior distributions in section 3.1, the optimal variational posterior distribution over latent variables $\tilde{q}(S, V|\boldsymbol{O}, m)$ can be obtained as:

$$
\tilde{q}(S, V|\boldsymbol{O}, m) \propto \prod_{t=1}^T \tilde{a}_{s^{t-1}s^t}\tilde{c}_{s^tv^t}\tilde{b}_{s^tv^t}(\boldsymbol{O}^t), \tag{6}
$$

where

$$\tilde{a}_{s^{t-1}s^t} = \exp\left\{\Psi(\tilde{\phi}_{s^{t-1}s^t}) - \Psi(\sum\nolimits_{s^{t'}=1}^{J} \tilde{\phi}_{s^{t-1}s^{t'}})\right\},$$

$$\tilde{c}_{s^t v^t} = \exp\left\{\Psi(\tilde{\varphi}_{s^t v^t}) - \Psi(\sum\nolimits_{v^{t'}=1}^{L} \tilde{\varphi}_{s^t v^{t'}})\right\},$$

$$\tilde{b}_{s^t v^t}(\boldsymbol{O}^t) = \exp\left\{ D/2\left(\log 2\pi - 1/\tilde{\xi}_{s^t v^t} + \Psi(\tilde{\eta}_{s^t v^t}/2)\right) - \right.$$
$$\left. -1/2\sum\nolimits_{d=1}^{D}\left(\log(\tilde{R}_{s^t v^t,d}/2) + (O_d^t - \tilde{\nu}_{s^t v^t,d})^2 \tilde{\eta}_{s^t v^t}/\tilde{R}_{s^t v^t,d}\right)\right\}. (7)$$

$\Psi(y)$ is a digamma function. From these results, transition and occupation probability $\tilde{\gamma}_{ij}^t$ and $\tilde{\zeta}_{ij}^t$ can be obtained by using either a deterministic assignment via the Viterbi algorithm or a probabilistic assignment via the Forward-Backward algorithm. Thus, $\tilde{q}(\Theta|\boldsymbol{O}, m)$ and $\tilde{q}(S, V|\boldsymbol{O}, m)$ can be calculated iteratively that result in maximizing $\mathcal{F}_m$.

## 4  VB training algorithm for acoustic models

Based on the discussion in section 3, a VB training algorithm for an acoustic model based on an HMM and Gaussian mixture model with a fixed model structure $m$ is as follows:

────────────────────────────────────────────

**Step 1)** Initialize $\tilde{\gamma}_{ij}^t[\tau = 0]$, $\tilde{\zeta}_{ij}^t[\tau = 0]$ and set $\Phi^0$.

**Step 2)** Compute $q(S, V|\boldsymbol{O}, m)[\tau + 1]$ using $\tilde{\gamma}_{ij}^t[\tau]$, $\tilde{\zeta}_{ij}^t[\tau]$ and $\Phi^0$.

**Step 3)** Update $\tilde{\gamma}_{ij}^t[\tau+1]$ and $\tilde{\zeta}_{ij}^t[\tau+1]$ using $q(S, V|\boldsymbol{O}, m)[\tau+1]$ via the Viterbi algorithm or Forward-Backward algorithm.

**Step 4)** Compute $\tilde{\Phi}[\tau + 1]$ using $\tilde{\gamma}_{ij}^t[\tau + 1]$, $\tilde{\zeta}_{ij}^t[\tau + 1]$ and $\Phi^0$.

**Step 5)** Compute $q(\Theta|\boldsymbol{O}, m)[\tau + 1]$ using $\tilde{\Phi}[\tau + 1]$ and calculate $\mathcal{F}_m[\tau]$ based on $q(\Theta|\boldsymbol{O}, m)[\tau + 1]$ and $q(S, V|\boldsymbol{O}, m)[\tau + 1]$.

**Step 6)** If $|(\mathcal{F}_m[\tau + 1] - \mathcal{F}_m[\tau])/F_m[\tau + 1]| \leq \varepsilon$, then stop; otherwise set $\tau \leftarrow \tau + 1$ and go to **Step 2**.

────────────────────────────────────────────

$\tau$ denotes an iteration count. In our experiments, we employed the Viterbi algorithm in **Step 3**.

## 5  Variational Bayesian estimation and clustering for speech recognition

In the previous section, we described a VB training algorithm for HMMs. Here, we explain VBEC, which constructs an acoustic model based on SST-HMMs and recognizes speech based on the Bayesian prediction classification. VBEC consists of three phases: model structure selection, retraining and recognition. The model structure is determined based on triphone-state clustering by using the phonetic decision tree method [4] [7]. The phonetic decision tree is a kind of binary tree that has a phonetic "Yes/No" question attached at each node, as shown in Figure 2. Let $\Omega(n)$ denote a set of states held by a tree node $n$. We start with only a root node ($n = 0$), which holds a set of all the triphone HMM states $\Omega(0)$ for an identical center phoneme. The set of triphone states is then split into two sets, $\Omega(n_Y)$ and $\Omega(n_N)$, which are held by two new nodes, $n_Y$ and $n_N$, respectively, as shown in Figure 3. The partition is determined by an answer to a phonetic question such as "is the preceding phoneme a vowel?" or "is the following phoneme a nasal?" We choose a particular question for a node that maximize the gain of $\mathcal{F}^m$ when the node is split into two

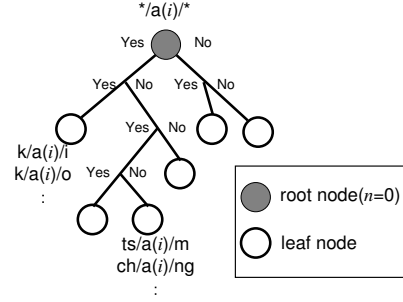

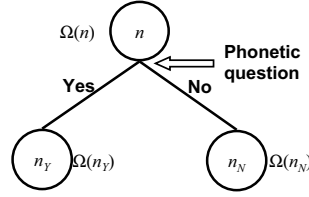

Figure 2: A set of all triphone HMM states */a(i)/* is clustered based on the phonetic decision tree method.

Figure 3: Splitting a set of triphone HMM states $\Omega(n)$ into two sets $\Omega(n_Y)$ $\Omega(n_N)$ by answering phonetic questions according to an objective function.

nodes, and if all the questions decrease $\mathcal{F}^m$ after splitting, we stop splitting. We continue this splitting successively for every new set of states to obtain a binary tree, each leaf node of which holds a clustered set of triphone states. The states belonging to the same cluster are merged into a single state. A set of triphones is thus represented by a set of shared-state triphone HMMs (SST-HMMs). An HMM, which represents a phonemic unit, usually consists of a linear sequence of three or four states. A decision tree is produced specifically for each state in the sequence, and the trees are independent of each other.

Note that in the triphone-states clustering mentioned above, we assume the following conditions to reduce computations:

- The state assignments while splitting are fixed.
- A single Gaussian distribution for one state is used.
- Contributions of the transition probabilities to the objective function are ignored.

By using these conditions, latent variables are removed. As a result, all variational posteriors and $\mathcal{F}_m$ can be obtained as closed forms without an iterative procedure.

Once we have obtained the model structure, we retrain the posterior distributions using the VB algorithm given in section 4. In recognition, an unknown datum $\boldsymbol{x}^t$ for a frame $t$ is classified as the optimal phoneme class $y$ using the predictive posterior classification probability $p(y|\boldsymbol{x}^t, \boldsymbol{O}, \tilde{m}) \equiv p(y)p(\boldsymbol{x}^t|y, \boldsymbol{O}, \tilde{m})/p(\boldsymbol{x}^t)$ for the estimated model structure $\tilde{m}$. Here, $p(y)$ is the class prior obtained by language and lexcon models, and $p(\boldsymbol{x}^t|y, \boldsymbol{O}, \tilde{m})$ is the predictive density. If we approximate the true posterior $p(\Theta|y, \boldsymbol{O}, \tilde{m})$ by the estimated variational posteriors $\tilde{q}(\Theta|y, \boldsymbol{O}, \tilde{m})$, $p(\boldsymbol{x}^t|y, \boldsymbol{O}, \tilde{m})$ can be calculated by $p(\boldsymbol{x}^t|y, \boldsymbol{O}, \tilde{m}) \approx \int p(\boldsymbol{x}^t|y, \Theta, \tilde{m})\tilde{q}(\Theta|y, \boldsymbol{O}, \tilde{m})d\Theta$. Therefore, the optimal class $y$ can be obtained by

$$y = \arg\max_{y'} p(y'|\boldsymbol{x}^t, \boldsymbol{O}, \tilde{m}) \approx \arg\max_{y'} p(y') \int p(\boldsymbol{x}^t|y', \Theta, \tilde{m})\tilde{q}(\Theta|y, \boldsymbol{O}, \tilde{m})d\Theta. \quad (8)$$

In the calculation of (8), the integral over Gaussian means and covariances for a frame can be solved analytically to be Student distributions. Therefore, we can compute a Bayesian predictive score for a frame, and then can compute a phoneme sequence score by using the Viterbi algorithm. Thus, we can construct a VBEC framework for speech recognition by selecting an appropriate model structure and estimating posterior distributions with the VB approach, and then obtaining a recognition result based on the Bayesian prediction classification.

| Table 1: Acoustic conditions | |
| --- | --- |
| Sampling rate | 16 kHz |
| Quantization | 16 bit |
| Feature vector | 12 - order MFCC |
| | with $\Delta$ MFCC |
| Window | Hamming |
| Frame size/shift | 25/10 ms |

| Table 2: Prepared HMM | |
| --- | --- |
| # of states | 3 (Left to right) |
| # of phoneme categories | 27 |
| Output distribution | Single Gaussian |

## 6  Experiments

We conducted two experiments to evaluate the effectiveness of VBEC. The first experiment compared VBEC with the conventional ML-BIC/MDL method for variable amounts of training data. In the ML-BIC/MDL, retraining and recognition are based on the ML approach and model structure selection is based on the BIC/MDL. The second experiment examined the robustness of the recognition performance with preset hyperparameter values against changes in the amounts of training data.

### 6.1  VBEC versus ML-BIC/MDL

The experimental conditions are summarized in Tables 1 and 2. As regards the hyperparameters, the mean and variance values of the Gaussian distribution were set at $\nu^0$ and $R^0$ in each root node, respectively, and the heuristics were removed for $\nu^0$ and $R^0$. The determination of $\xi^0$ and $\eta^0$ was still heuristic. We set $\xi^0 = \eta^0 = 0.01$, each of which were determined experimentally. The training and recognition data used in these experiments are shown in Table 3.

The total training data consisted of about 3,000 Japanese sentences spoken by 30 males. These sentences were designed so that the phonemic balance was maintained. The total recognition data consisted of 2,500 Japanese city names spoken by 25 males. Several subsets were randomly extracted from the training data set, and each subset was used to construct a set of SST-HMMs. As a result, 40 sets of SST-HMMs were prepared for several subsets of training data.

Figures 4 and 5 show the recognition rate and the total number of states in a set of SST-HMMs, according to the varying amounts of training data. As shown in Figure 4, when the number of training sentences was less than 40, VBEC greatly outperformed the ML-BIC/MDL (A). With ML-BIC/MDL (A), an appropriate model structure was obtained by maximizing an objective function $l_m^{BIC/MDL}$ w.r.t. $m$ based on BIC/MDL defined as:

$$l_m^{BIC/MDL} = l(\boldsymbol{O}, m) - \frac{\#(\Theta_\Omega)}{2} \log T_{\Omega(0)}, \qquad (9)$$

where, $l(\boldsymbol{O}, m)$ denotes the likelihood of training data $\boldsymbol{O}$ for a model structure $m$, $\#(\Theta_\Omega)$ denotes the number of free parameters for a set of states $\Omega$, and $T_{\Omega(0)}$ denotes the total frame number of training data for a set of states $\Omega(0)$ in a root node, as shown in Figure 2. The term $\frac{\#(\Theta_\Omega)}{2} \log T_{\Omega(0)}$ in Eq.(9) is regarded as a penalty term added to a likelihood, and is dependent on the number of free parameters $\#(\Theta_\Omega)$ and total frame number $T_{\Omega(0)}$ of the training data. ML-BIC/MDL (A) was based on the original definitions of BIC/MDL and has been widely used in speech recognition [2] [5]. With such small amounts of training data, there was a great difference between the total number of shared states with VBEC and

Table 3: Training and recognition data

| Training | Continuous speech sentences (Acoustical Society of Japan) |
| --- | --- |
| Recognition | 100 city names (Japan Electronic Industry Development Association) |

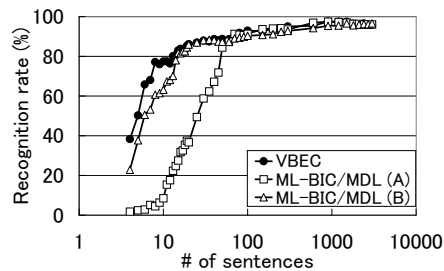
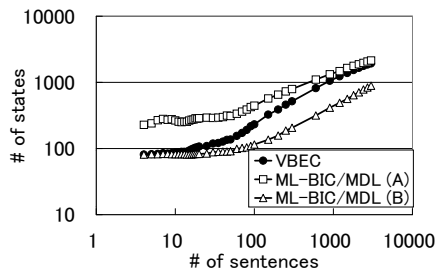

Figure 4: Recognition rates according to the amounts of training data based on the VBEC and ML-BIC/MDL (A) and (B). The horizontal axis is scaled logarithmically.

Figure 5: Number of shared states according to the amounts of training data based on the VBEC and ML-BIC/MDL (A) and (B). The horizontal and vertical axes are scaled logarithmically.

ML-BIC/MDL (A) (Figure 5). This suggests that VBEC, which does not use an asymptotic assumption, determines the model structure more appropriately than the ML-BIC/MDL (A), when the training data size is small.

Next, we adjusted the penalty term of ML-BIC/MDL in Eq. (9) so that the total numbers of states for small amounts of data were as close as possible to those of VBEC (ML-BIC/MDL (B) in Figure 5). Nevertheless, the recognition rates obtained by VBEC were about 15 % better than those of ML-BIC/MDL (B) with fewer than 15 training sentences (Figure 4). With such very small amounts of data, the VBEC and ML-BIC/MDL (B) model structures were almost same (Figure 5). It is assumed that the effects of the posterior estimation and the Bayesian prediction classification (Eq. (8)) suppressed the over-fitting of the models to very small amounts of training data compared with the ML estimation and recognition in ML-BIC/MDL (B).

With more than 100 training sentences, the recognition rates obtained by VBEC converged asymptotically to those obtained by ML-BIC/MDL methods as the amounts of training data became large.

In summary, VBEC performed as well or better for every amount of training data. This advantage was due to the superior properties of VBEC, e.g., the appropriate determination of the number of states and the suppression effect on over-fitting.

### 6.2 Influence of hyperparameter values on the quality of SST-HMMs

Throughout the construction of the model structure, the estimation of the posterior distribution, and recognition, we used a fixed combination of hyperparameter values, $\xi^0 = \eta^0 = 0.01$. In the small-scale experiments conducted in previous research [1] [3] [6] [8], the selection of such values was not a major concern. However, when the scale of the target application is large, the selection of hyperparameter values might affect the quality of the models. Namely, the best or better values might differ greatly according to the amounts of training data. Moreover, estimating appropriate hyperparameters with training SST-HMMs needs so much time that it is impractical in speech recognition. Therefore, we examined how robustly the SST-HMMs produced by VBEC performed against changes in the hyperparameter values with varying amounts of training data.

We varied the values of hyperparameters $\xi^0$ and $\eta^0$ from 0.0001 to 1, and examined the speech recognition rates in two typical cases; one in which the amount of data was very small (10 sentences) and one in which the amount was fairly large (150 sentences). Tables

Table 4: Recognition rates in each prior distribution parameter when using training data of 10 sentences.

| $\xi^0$ | $\eta^0$ | | | | |
|---|---|---|---|---|---|
| | $10^0$ | $10^{-1}$ | $10^{-2}$ | $10^{-3}$ | $10^{-4}$ |
| $10^0$ | 1.0 | **66.3** | 65.9 | **66.5** | 66.1 |
| $10^{-1}$ | 2.2 | 65.9 | **66.2** | **66.7** | 66.1 |
| $10^{-2}$ | 31.2 | 66.1 | **66.5** | **66.3** | 65.5 |
| $10^{-3}$ | 60.3 | **66.2** | **66.7** | 66.1 | 65.5 |
| $10^{-4}$ | **66.5** | **66.6** | **66.3** | 65.5 | 64.6 |

Table 5: Recognition rates in each prior distribution parameter when using training data of 150 sentences.

| $\xi^0$ | $\eta^0$ | | | | |
|---|---|---|---|---|---|
| | $10^0$ | $10^{-1}$ | $10^{-2}$ | $10^{-3}$ | $10^{-4}$ |
| $10^0$ | 22.0 | **93.5** | **94.0** | **93.1** | 92.3 |
| $10^{-1}$ | 49.3 | **94.3** | **93.9** | **93.3** | 92.5 |
| $10^{-2}$ | 83.5 | **94.4** | **93.2** | 92.3 | 92.3 |
| $10^{-3}$ | 92.5 | **93.8** | **93.3** | 92.5 | 92.4 |
| $10^{-4}$ | **94.1** | **93.2** | 92.3 | 92.3 | 92.2 |

4 and 5 show the recognition rates for each combination of hyperparameters. We can see that the hyperparameter values for acceptable performance are broadly distributed for both very small and fairly large amounts of training data. Moreover, roughly the ten best recognition rates are highlighted in the tables. The combinations of hyperparameter values that achieved the highlighted recognition rates were similar for the two different amounts of training data. Namely, appropriate combinations of hyperparameter values can consistently provide good performance levels regardless of the varying amounts of training data.

In summary, the hyperparameter values do not greatly influence the quality of the SST-HMMs. This suggests that it is not necessary to select the hyperparameter values very carefully.

## 7   Conclusion

In this paper, we proposed VBEC, which constructs SST-HMMs based on the VB approach, and recognizes speech based on the Bayesian prediction classification. With VBEC, the model structure of SST-HMMs is adaptively determined according to the amounts of given training data, and therefore a robust speech recognition system can be constructed. The first experimental results, obtained by using real speech recognition tasks, showed the effectiveness of VBEC. In particular, when the training data size was small, VBEC significantly outperformed conventional methods. The second experimental results suggested that it is not necessary to select the hyperparameter values very carefully. From these results, we conclude that VBEC provides a completely Bayesian framework for speech recognition which effectively hundles the sparse data problem.

## Footnotes

[1]These criteria have been independently proposed, but they are practically the same. Therefore, we refer to them hereafter as BIC/MDL.

## References

[1] H. Attias, "A Variational Bayesian Framework for Graphical Models," *NIPS12, MIT Press*, (2000).

[2] W. Chou and W. Reichl, "Decision Tree State Tying Based on Penalized Bayesian Information Criterion," *Proc. ICASSP'99*, vol. 1, pp. 345-348, (1999).

[3] Z. Ghahramani and M. J. Beal, "Variational Inference for Bayesian Mixtures of Factor Analyzers," *NIPS12, MIT Press*, (2000).

[4] J. J. Odell, "The Use of Context in Large Vocabulary Speech Recognition," *PhD thesis, Cambridge University*, (1995).

[5] K. Shinoda and T. Watanabe, "Acoustic Modeling Based on the MDL Principle for Speech Recognition," *Proc. EuroSpeech'97*, vol. 1, pp. 99-102, (1997).

[6] N. Ueda and Z. Ghahramani, "Bayesian Model Search for Mixture Models Based on Optimizing Variational Bounds," *Neural Networks*, vol. 15, pp. 1223-1241, (2002).

[7] S. Watanabe et. al., "Constructing Shared-State Hidden Markov Models Based on a Bayesian Approach," *Proc. ICSLP'02,* vol. 4, pp. 2669-2672, (2002).

[8] S. Waterhouse et. al., "Bayesian Methods for Mixture of Experts," *NIPS8, MIT Press*, (1995).
